# Predicting the Geometry of Metal Binding Sites from Protein Sequence

**Paolo Frasconi**
Università degli Studi di Firenze
Via di S. Marta 3, 50139 Firenze, Italy
p-f@dsi.unifi.it

**Andrea Passerini**
Università degli Studi di Trento
Via Sommarive, 14, 38100 Povo, Italy
passerini@disi.unitn.it

## Abstract

Metal binding is important for the structural and functional characterization of proteins. Previous prediction efforts have only focused on bonding state, i.e. deciding which protein residues act as metal ligands in some binding site. Identifying the geometry of metal-binding sites, i.e. deciding which residues are jointly involved in the coordination of a metal ion is a new prediction problem that has been never attempted before from protein sequence alone. In this paper, we formulate it in the framework of learning with structured outputs. Our solution relies on the fact that, from a graph theoretical perspective, metal binding has the algebraic properties of a matroid, enabling the application of greedy algorithms for learning structured outputs. On a data set of 199 non-redundant metalloproteins, we obtained precision/recall levels of 75%/46% correct ligand-ion assignments, which improves to 88%/88% in the setting where the metal binding state is known.

## 1 Introduction

Metal ions play important roles in protein function and structure and metalloproteins are involved in a number of diseases for which medicine is still seeking effective treatment, including cancer, Parkinson, dementia, and AIDS [10]. A metal binding site typically consists of an ion bound to one or more protein residues (called ligands). In some cases, the ion is embedded in a prosthetic group (e.g. in the case of heme). Among the 20 amino acids, the four most common ligands are cysteine (C), histidine (H), aspartic acid (D), and glutamic acid (E). Highly conserved residues are more likely to be involved in the coordination of a metal ion, although in the case of cysteines, conservation is also often associated with the presence of a disulfide bridge (a covalent bond between the sulfur atoms of two cysteines) [8]. Predicting metal binding from sequence alone can be very useful in genomic annotation for characterizing the function and the structure of non determined proteins, but also during the experimental determination of new metalloproteins. Current high-throughput experimental technologies only annotate whole proteins as metal binding [13], but cannot determine the involved ligands. Most of the research for understanding metal binding has focused on finding sequence patterns that characterize binding sites [8]. Machine learning techniques have been applied only more recently.

The easiest task to formulate in this context is bonding state prediction, which is a binary classification problem: either a residue is involved in the coordination of a metal ion or is free (in the case of cysteines, a third class can also be introduced for disulfide bridges). This prediction task has been addressed in a number of recent works in the case of cysteines only [6], in the case of transition metals (for C and H residues) [12] and for in the special but important case of zinc proteins (for C,H,D, and E residues) [11, 14]. Hovever, classification of individual residues does not provide sufficient information about a binding site. Many proteins bind to several ions in their holo form and a complete characterization requires us to identify the site geometry, i.e. the tuple of residues coordinating each individual ion. This problem has been only studied assuming knowledge of the protein 3D structure (e.g. [5, 1]), limiting its applicability to structurally determined proteins or their

close homologs, but not from sequence alone. Abstracting away the biology, this is a structured output prediction problem where the input consists of a string of protein residues and the output is a labeling of each residue with the corresponding ion identifier (specific details are given in the next section).

The supervised learning problem with structured outputs has recently received a considerable amount of attention (see [2] for an overview). The common idea behind most methods consists of learning a function $F(x, y)$ on input-output pairs $(x, y)$ and, during prediction, searching the argument $y$ that maximises $F$ when paired with the query input $x$. The main difficulty is that the search space on which $y$ can take values has usually exponential size (in the length of the query). Different structured output learners deal with this issue by exploiting specific domain properties for the application at hand. Some researchers have proposed probabilistic modeling and efficient dynamic programming algorithms (e.g. [16]). Others have proposed large margin approaches combined with clever algorithmic ideas for reducing the number of constraints (e.g. [15] in the case of graph matching). Another solution is to construct the structured output in a suitable Hilbert space of features and seek the corresponding pre-image for obtaining the desired discrete structure [17]. Yet another is to rely on a state-space search procedure and learn from examples good moves leading to the desired goal [4].

In this paper we develop a large margin solution that does not require a generative model for producing outputs. We borrow ideas from [15] and [4] but specifically take advantage of the fact that, from a graph theoretical perspective, the metal binding problem has the algebraic structure of a matroid, enabling the application of greedy algorithms.

## 2   A formalization of the metal binding sites prediction problem

A protein sequence $s$ is a string in the alphabet of the 20 amino acids. Since only some of the 20 amino acids that exist in nature can act as ligands, we begin by extracting from $s$ the subsequence $x$ obtained by deleting characters corresponding to amino acids that never (or very rarely) act as ligands. By using $\mathcal{T} = \{C, H, D, E\}$ as the set of candidate ligands, we cover 92% ligands of structurally known proteins. A large number of interesting cases (74% in transition metals) is covered by just considering cysteines and histidines, i.e. $\mathcal{T} = \{C, H\}$. We also introduce the set $\mathcal{I}$ of symbols associated with metal ion identifiers. $\mathcal{I}$ includes the special *nil* symbol. The goal is to predict the *coordination* relation between amino acids in $x$ and metal ions identifiers in $\mathcal{I}$. Amino acids that are not metal-bound are linked to *nil*. Ideally, it would be also interesting to predict the chemical element of the bound metal ion. However, previous studies suggest that distinguishing the chemical element from sequence alone is a difficult task [12]. Hence, ion identifiers will have no chemical element attribute attached. In practice, we fix a maximum number $m$ of possible ions ($m = 4$ in the subsequent experiments, covering 93% of structurally known proteins) and let $\mathcal{I} = \{nil, \iota_1, \ldots, \iota_m\}$.

The number of admissible binding geometries for a given protein chain having $n$ candidate ligands is the multinomial coefficient $\frac{n!}{k_1! k_2! \cdots k_m! (n - k_1 - \cdots - k_m)!}$ being $m$ the number of ions and $k_i$ the number of ligands for ion $\iota_i$. In practice, each ion is coordinated by a variable number of ligands (typically ranging from 1 to 4, but occasionally more), and each protein chain binds a variable number of ions (typically ranging from 1 to 4). The number of candidate ligands $n$ grows linearly with the protein chain. For example, in the case of PDB chain 1H0Hb (see Figure 1), there are $n = 52$ candidate ligands and $m = 3$ ions coordinated by 4 residues each, yielding a set of $7 \cdot 10^{15}$ admissible conformations.

It is convenient to formulate the problem in a graph theoretical setting. In this view, the string $x$ should be regarded as a set of vertices labeled with the corresponding amino acid in $\mathcal{T}$. The semantic of $x$ will be clear from the context and for simplicity we will avoid additional notation.

**Definition 2.1** (MBG property). Let $x$ and $\mathcal{I}$ be two sets of vertices (associated with candidate ligands and metal ion identifiers, respectively). We say that a bipartite edge set $y \subset x \times \mathcal{I}$ satisfies the metal binding geometry (MBG) property if the degree of each vertex in $x$ in the graph $(x \cup \mathcal{I}, y)$ is at most 1.

For a given $x$, let $\mathcal{Y}_x$ denote the set of $y$ that satisfy the MBG property. Let $F_x : \mathcal{Y}_x \mapsto \mathbb{R}^+$ be a function that assigns a positive score to each bipartite edge set in $\mathcal{Y}_x$. The MBG problem consists of finding $\arg \max_{y \in \mathcal{Y}_x} F_x(y)$.

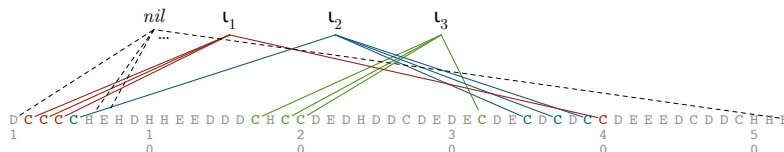

Figure 1: Metal binding structure of PDB entry 1H0Hb. For readability, only a few connections from free residues to the *nil* symbol are shown.

Note that the MBG problem is not a matching problem (such as those studied in [15]) since more than one edge can be incident to vertices belonging to $\mathcal{I}$. As discussed above, we are not interested in distinguishing metal ions based on the element type. Hence, any two label-isomorphic bipartite graphs (obtained by exchanging two non-nil metal ion vertices) should be regarded as equivalent. Outputs $y$ should be therefore regarded as equivalence classes of structures (in the 1H0Hb example above, there are $7 \cdot 10^{15}/3!$ equivalence classes, each corresponding to a permutation of $\iota_1, \iota_2, \iota_3$). For simplicity, we will slightly abuse notation and avoid this distinction in the following.

We could also look over the MBG problem by analogy with language parsing using formal grammars. In this view, the binding geometry consists of a very shallow "parse tree" for string $x$, as examplified in Figure 1. A difficulty that is immediately apparent is that the underlying grammar needs to be *context sensitive* in order to capture the crossing-dependencies between bound amino acids. In real data, when representing metal bonding state in this way, crossing edges are very common. This view enlightens a difficulty that would be encountered by attempting to solve the structured output problem with a generative model as in [16].

# 3   A greedy algorithm for constructing structured outputs

The core idea of the solution used in this paper is to avoid a generative model as a component of the structured output learner and cast the construction of an output structure into a maximum weight problem that can be solved by a greedy algorithm.

**Definition 3.1** (Matroid). A matroid (see e.g. [9]) is an algebraic structure $\mathcal{M} = (S, \mathcal{Y})$ where $S$ is a finite set and $\mathcal{Y}$ a family of subsets of $S$ such that: i) $\emptyset \subseteq \mathcal{Y}$; ii) all proper subsets of a set $y$ in $\mathcal{Y}$ are in $\mathcal{Y}$; iii) if $y$ and $y'$ are in $\mathcal{Y}$ and $|y| < |y'|$ then there exists $e \in y' \setminus y$ such that $y \cup \{e\} \in \mathcal{Y}$.

Elements of $\mathcal{Y}$ are called *independent sets*. If $y$ is an independent set, then $\text{ext}(y) = \{e \in S : y \cup \{e\} \in \mathcal{Y}\}$ is called the extension set of $y$. A maximal (having an empty extension set) independent set is called a *base*. In a *weighted* matroid, a local weight function $v : S \mapsto \mathbb{R}^+$ assigns a positive number $v(e)$ to each element $e \in S$. The weight function allows us to compare two structures in the following sense. A set $y = \{e_1, \ldots, e_n\}$ is lexicographically greater than set $y'$ if its monotonically decreasing sequence of weights $(v(e_1), \ldots, v(e_n))$ is lexicographically greater than the corresponding sequence for $y'$. The following classic result (see e.g. [9]) is the underlying support for many greedy algorithms:

**Theorem 3.2** (Rado 1957; Edmonds 1971). *For any nonnegative weighting over $S$, a lexicographically maximum base in $\mathcal{Y}$ maximizes the global objective function $F(y) = \sum_{e \in y} v(e)$.*

Weighted matroids can be seen as a kind of discrete counterparts of concave functions: thanks to the above theorem, if $\mathcal{M}$ is a weighted matroid, then the following greedy algorithm is guaranteed to find the optimal structure, i.e. $\arg\max_{y \in \mathcal{Y}} F(y)$:

GREEDYCONSTRUCT$(\mathcal{M}, F)$

$\quad y \leftarrow \emptyset$
$\quad$**while** $\text{ext}(y) \neq \emptyset$
$\quad\quad$**do** $y \leftarrow y \cup \left\{ \arg\max_{e \in \text{ext}(y)} F(y \cup \{e\}) \right\}$
$\quad$**return** $y$

This theory shows that if the structured output space being searched satisfies the property of a matroid, learning structured outputs may be cast into the problem of learning the objective function

$F$ for the greedy algorithm. When following this strategy, however, we may perceive the additive form of $F$ as a strong limitation as it would prescribe to predict $v(e)$ independently for each part $e \in S$, while the whole point of structured output learning is to end-up with a *collective* decision about which parts should be present in the output structure. But interestingly, the additive form of the objective function as in Theorem 3.2 is not a necessary condition for the greedy optimality of matroids. In facts, Helman et al. [7] show that the classic theory can be generalized to so-called *consistent* objective functions, i.e. functions that satisfy the following additional constraints:

$$F(y \cup \{e\}) \geq F(y \cup \{e'\}) \Rightarrow F(y' \cup \{e\}) \geq F(y' \cup \{e'\}) \tag{1}$$

for any $y \subset y' \subset S$ and $e, e' \in S \setminus y'$.

**Theorem 3.3** (Helman et al. 1993). *If $F$ is a consistent objective function then, for each matroid on $S$, all greedy bases are optimal.*

Note that the sufficient condition of Theorem 3.3 is also necessary for a slighly more general class of algebraic structures that include matroids, called *matroid embeddings* [7]. We now show that the MBG problem is a suitable candidate for a greedy algorithmic solution.

**Theorem 3.4.** *If each $y \in \mathcal{Y}_x$ satisfies the MBG property, then $\mathcal{M}_x = (S_x, \mathcal{Y}_x)$ is a matroid.*

*Proof.* Suppose $y' \in \mathcal{Y}_x$ and $y \subseteq y'$. Removing an edge from $y'$ cannot increase the degree of any vertex in the bipartite graph so $y \in \mathcal{Y}_x$. Also, suppose $y \in \mathcal{Y}_x$, $y' \in \mathcal{Y}_x$, and $|y| < |y'|$. Then there must be at least one vertex $t$ in $x$ having no incident edges in $y$ and such that $(\iota, t) \in y'$ for some $\iota \in \mathcal{I}$. Therefore $y \cup \{(\iota, t)\}$ also satisfies the MBG property and belongs to $\mathcal{Y}_x$, showing that $\mathcal{M}_x$ is a matroid. $\square$

We can finally formulate the greedy algorithm for constructing the structured output in the MBG problem. Given the input $x$, we begin by forming the associated MBG matroid $M_x$ and a corresponding objective function $F_x : \mathcal{Y}_x \mapsto I\!\!R^+$ (in the next section we will show how to learn the objective function from data). The output structure associated with $x$ is then computed as

$$f(x) = \arg\max_{y \in \mathcal{Y}_x} F_x(y) = \text{GREEDYCONSTRUCT}(\mathcal{M}_x, F_x). \tag{2}$$

The following result immediately follows from Definition 2.1 and Theorem 3.3:

**Corollary 3.5.** *Let $(x, y)$ be an MBG instance. If $F_x$ is a consistent objective function and $F_x(y' \cup \{e\}) > F_x(y' \cup \{e'\})$ for each $y' \subset y$, $e \in \text{ext}(y') \cap y$ and $e' \in \text{ext}(y') \setminus y$, then $\text{GREEDYCONSTRUCT}((S_x, \mathcal{Y}_x), F_x)$ returns $y$.*

## 4   Learning the greedy objective function

A data set for the MBG problem consist of pairs $\mathcal{D} = \{(x_i, y_i)\}$ where $x_i$ is a string in $\mathcal{T}^*$ and $y_i$ a bipartite graph. Corollary 3.5 directly suggests the kind of constraints that the objective function needs to satisfy in order to minimize the empirical error of the structured-output problem. For any input string $x$ and (partial) output structure $y \in \mathcal{Y}$, let $F_x(y) = w^T \phi_x(y)$, being $w$ a weight vector and $\phi_x(y)$ a feature vector for $(x, y)$. The corresponding max-margin formulation is

$$\min \frac{1}{2} \|w\|^2 \tag{3}$$

$$\text{subject to:} \quad w^T \left( \phi_{x_i}(y' \cup \{e\}) - \phi_{x_i}(y' \cup \{e'\}) \right) \geq 1 \tag{4}$$

$$w^T \left( \phi_{x_i}(y'' \cup \{e\}) - \phi_{x_i}(y'' \cup \{e'\}) \right) \geq 1 \tag{5}$$

$$\forall i = 1, \ldots, |\mathcal{D}|, \ \forall y' \subset y_i, \ \forall e \in \text{ext}(y') \cap y_i, \ \forall e' \in \text{ext}(y') \setminus y_i,$$
$$\forall y'' : y' \subset y'' \subset S_x.$$

Intuitively, the first set of constraints (Eq. 4) ensures that "correct" extensions (i.e. edges that actually belong to the target output structure $y_i$) receive a higher weight than "wrong" extensions (i.e. edges that do not belong to the target output structure). The purpose of the second set of constraints (Eq. 5) is to force the learned objective function to obey the consistency property of Eq. (1), which in turns ensures the correctness of the greedy algorithm thanks to Theorem 3.3. As usual, a regularized

variant with soft constraints can be formulated by introducing positive slack variables and adding their 1-norm times a regularization coefficient to Eq. (3). The number of resulting constraints in the above formulation grows exponentially with the number of edges in each example, hence naively solving problem (3–5) is practically unfeasible. However, we can seek an approximate solution by leveraging the efficiency of the greedy algorithm also *during* learning. For this purpose, we will use an online active learner that samples constraints chosen by the execution of the greedy construction algorithm.

For each epoch, the algorithm maintains the current highest scoring partial correct output $y_i' \subseteq y_i$ for each example, initialized with the empty MBG structure, where the score is computed by the current objective function $F$. While there are "unprocessed" examples in $\mathcal{D}$, the algorithm picks a random one and its current best MBG structure $y'$. If there are no more correct extensions of $y'$, then $y' = y_i$ and the example is removed from $\mathcal{D}$. Otherwise, the algorithm evaluates each correct extension of $y'$, updates the current best MBG structure, and invokes the online learner by calling FORCE-CONSTRAINT, which adds a constraint derived from a random incorrect extension (see Eq. 4). It also performs a predefined number $L$ of lookaheads by picking a random superset of $y''$ which is included in the target $y_i$, evaluating it and updating the best MBG structure if needed, and adding a corresponding consistency constraint (see Eq. 5). The epoch terminates when all examples are processed. In practice, we found that a single epoch over the data set is sufficient for convergence. Pseudocode for one epoch is given below.

GREEDYEPOCH($\mathcal{D}, L$)
    **for** $i \leftarrow 1, \ldots, |\mathcal{D}|$
        **do** $y_i' \leftarrow \emptyset$
    **while** $\mathcal{D} \neq \emptyset$
        **do** pick a random example $(x_i, y_i) \in \mathcal{D}$
            $y' \leftarrow y_i', \; y_i' \leftarrow \emptyset$
            **if** $\text{ext}(y') \cap y_i = \emptyset$
                **then** $\mathcal{D} \leftarrow \mathcal{D} \setminus (x_i, y_i)$
                **else** **for** each $e \in \text{ext}(y') \cap y_i$
                      **do** pick randomly $e' \in \text{ext}(y') \setminus y_i$
                          **if** $F(y_i') < F(y' \cup \{e\})$ **then** $y_i' \leftarrow y' \cup \{e\}$
                          FORCE-CONSTRAINT($F_{x_i}(y' \cup \{e\}) - F_{x_i}(y' \cup \{e'\}) \geq 1$)
                          **for** $l \leftarrow 1, \ldots, L$
                              **do** randomly choose $y'' : y' \subset y'' \subset y_i \wedge e, e' \in S_x \setminus y''$
                                  FORCE-CONSTRAINT($F_{x_i}(y'' \cup \{e\}) - F_{x_i}(y'' \cup \{e'\}) \geq 1$)
                                **if** $F(y_i') < F(y'' \cup \{e\})$ **then** $y_i' \leftarrow y'' \cup \{e\}$

There are several suitable online learners implementing the interface required by the above procedure. Possible candidates include perceptron-like or ALMA-like update rules like those proposed in [4] for structured output learning (in our case the update would depend on the difference between feature vectors of correctly and incorrectly extended structures in the inner loop of GREEDYEPOCH). An alternative online learner is the LaSVM algorithm [3] equipped with obvious modifications for handling constraints between pairs of examples. LaSVM is an SMO-like solver for the *dual* version of problem (3–5) that optimizes one or two coordinates at a time, alternating *process* (on newly acquired examples, generated in our case by the FORCE-CONSTRAINT procedure) and *reprocess* (on previously seen support vectors or patterns) steps. The ability to work efficiently in the dual is the most appealing feature of LaSVM in the present context and advantageous with respect to perceptron-like approaches. Our unsuccessful preliminary experiments with simple feature vectors confirmed the necessity of flexible design choices for developing rich feature spaces. Kernel methods are clearly more attractive in this case. We will therefore rewrite the objective function $F$ using a kernel $k(z, z') = \langle \phi_x(y), \phi_{x'}(y') \rangle$ between two structured instances $z = (x, y)$ and $z' = (x', y')$, so that $F_x(y) = F(z) = \sum_i \alpha_i k(z, z_i)$.

Let $\sigma_i(z)$ denote the set of edges incident on ion $\iota_i \in \mathcal{I} \setminus nil$ and $n(z)$ the number of non-nil ion identifiers that have at least one incident edge. Below is a top-down definition of the kernel used in

the subsequent experiments.

$$k(z, z') = k_{\text{glob}}(z, z') \sum_{i=1}^{n(z)} \sum_{j=1}^{n(z')} \frac{k_{\text{mbs}}(\sigma_i(z), \sigma_j(z'))}{n(z)n(z')} \qquad (6)$$

$$k_{\text{glob}}(z, z') = \delta(n(z), n(z')) \frac{2 \min\{|x|, |x'|\}}{|x| + |x'|} \qquad (7)$$

$$k_{\text{mbs}}(\sigma_i(z), \sigma_j(z')) = \delta(|\sigma_i(z)|, |\sigma_j(z')|) \sum_{\ell=1}^{|\sigma_i(z)|} k_{\text{res}}(x_i(\ell), x'_j(\ell)) \qquad (8)$$

where $\delta(a, b) = 1$ iff $a = b$, $x_i(\ell)$ denotes the $\ell$-th residue in $\sigma_i(z)$, taken in increasing order of sequential position in the protein, and $k_{\text{res}}(x_i(\ell), x'_j(\ell))$ is simply the dot product between the feature vectors describing residues $x_i(\ell)$ and $x'_j(\ell)$ (details on these features are given in Section 5). $k_{\text{mbs}}$ measures the similarity between individual sites (two sites are orthogonal if have a different number of ligands, a choice that is supported by protein functional considerations). $k_{\text{glob}}$ ensures that two structures are orthogonal unless they have the same number of sites and down weights their similarity when their number of candidate ligands differs.

## 5   Experiments

We tested the method on a dataset of non-redundant proteins previously used in [12] for metal bonding state prediction (`http://www.dsi.unifi.it/~passe/datasets/mbs06/dataset.tgz`). Proteins that do not bind metal ions (used in [12] as negative examples) are of no interest in the present case and were removed, resulting in a set of 199 metalloproteins binding transition metals. Following [12], we used $\mathcal{T} = \{C, H\}$ as the set of candidate ligands. Protein sequences were enriched with evolutionary information derived from multiple alignments. Profiles were obtained by running one iteration of PSI-BLAST on the non-redundant (nr) NCBI dataset, with an e-value cutoff of 0.005. Each candidate ligand $x_i(\ell)$ was described by a feature vector of 221 real numbers. The first 220 attributes consist of multiple alignment profiles in the window of 11 amino acids centered around $x_i(\ell)$ (the window was formed from the original protein sequence, not the substring $x_i$ of candidate ligands). The last attribute is the normalized sequence separation between $x_i(\ell)$ and $x_i(\ell - 1)$, using the N-terminus of the chain for $\ell = 1$.

A modified version of LaSVM (`http://leon.bottou.org/projects/lasvm`) was run with constraints produced by the GREEDYEPOCH procedure of Section 4, using a fixed regularization parameter $C = 1$, and $L \in \{0, 5, 10\}$. All experiments were repeated 30 times, randomly splitting the data into a training and test set in a ratio of 80/20. Two prediction tasks were considered, from unknown and from known metal bonding state (a similar distinction is also customary for the related task of disulfide bonds prediction, see e.g. [15]). In the latter case, the input $x$ only contains actual ligands and no $nil$ symbol is needed.

Several measures of performance are reported in Table 1. $P_E$ and $R_E$ are the precision and recall for the correct assignment between a residue and the metal ion identifier (ratio of correctly predicted coordinations to the number of predicted/actual coordinations); correct links to the $nil$ ion (that would optimistically bias the results) are ignored in these measures. $A_G$ is the geometry accuracy, i.e. the fraction of chains that are entirely correctly predicted. $P_S$ and $R_S$ are the metal binding site precision and recall, respectively (ratio of correctly predicted sites to the number of predicted/actual sites). Finally, $P_B$ and $R_B$ are precision and recall for metal bonding state prediction (as in binary classification, being "bonded" the positive class). Table 2 reports the breakdown of these performance measures for proteins binding different numbers of metal ions (for $L = 10$).

Results show that enforcing consistency constraints tends to improve recall, especially for the bonding state prediction, i.e. helps the predictor to assign a residue to a metal ion identifier rather than to $nil$. However, it only marginally improves precision and recall at the site level. Correct prediction of whole sites is very challenging and correct prediction of whole chains even more difficult (given the enormous number of alternatives to be compared). Hence, it is not surprising that some of these performance indicators are low. By comparison, absolute figures are not high even for the much easier task of disulfide bonds prediction [15]. Correct edge assignment, however, appears satisfactory and reasonably good when the bonding state is given. The complete experimental environment can be obtained from `http://www.disi.unitn.it/~passerini/nips08.tgz`.

Table 1: Experimental results.

ab-initio

| $L$ | $P_E$ | $R_E$ | $A_G$ | $P_S$ | $R_S$ | $P_B$ | $R_B$ |
|---|---|---|---|---|---|---|---|
| 0 | 75±5 | 46±5 | 12±4 | 18±6 | 14±6 | 81±5 | 51±6 |
| 5 | 66±5 | 52±4 | 14±6 | 20±7 | 17±6 | 79±4 | 64±6 |
| 10 | 63±5 | 52±5 | 13±6 | 20±7 | 15±6 | 78±4 | 68±5 |

metal bonding state given

| $L$ | $P_E$ | $R_E$ | $A_G$ | $P_S$ | $R_S$ |
|---|---|---|---|---|---|
| 0 | 87±2 | 87±2 | 64±6 | 65±6 | 65±6 |
| 5 | 87±3 | 87±3 | 65±7 | 66±7 | 66±7 |
| 10 | 88±3 | 88±3 | 67±7 | 67±7 | 67±7 |

Table 2: Breakdown by number of sites each chain. BS= (K)nown/(U)nknown bonding state.

| | # sites = 1 (132 chains) | | | | | # sites = 2 (48 chains) | | | | |
|---|---|---|---|---|---|---|---|---|---|---|
| BS | $P_E$ | $R_E$ | $P_S$ | $R_S$ | $A_G$ | $P_E$ | $R_E$ | $P_S$ | $R_S$ | $A_G$ |
| U | 62±6 | 57±6 | 25±9 | 21±8 | 19±8 | 67±9 | 46±8 | 14±12 | 6±8 | 3±6 |
| K | 97±2 | 97±2 | 92±6 | 92±6 | 92±6 | 73±5 | 73±5 | 21±10 | 21±10 | 20±11 |

| | # sites = 3 (11 chains) | | | | | # sites = 4 (8 chains) | | | | |
|---|---|---|---|---|---|---|---|---|---|---|
| BS | $P_E$ | $R_E$ | $P_S$ | $R_S$ | $A_G$ | $P_E$ | $R_E$ | $P_S$ | $R_S$ | $A_G$ |
| U | 65±16 | 33±13 | 1±5 | 1±5 | 0 | 44±31 | 24±20 | 3±11 | 2±6 | 0 |
| K | 61±12 | 61±12 | 8±11 | 9±13 | 0 | 37±25 | 37±25 | 1±2 | 1±2 | 0 |

# 6 Related works

As mentioned in the Introduction, methods for structured outputs usually learn a function $F$ on input-output pairs $(x, y)$ and construct the predicted output as $f(x) = \arg\max_y F(x, y)$. Our approach follows the same general principle.

There is a notable analogy between the constrained optimization problem (3–5) and the set of constraints derived in [15] for the related problem of disulfide connectivity. As in [15], our method is based on a large-margin approach for solving a structured output prediction problem. The underlying formal problems are however very different and require different algorithmic solutions. Disulfide connectivity is a (perfect) matching problem since each cysteine is bound to exactly one other cysteine (assuming known bonding state, yielding a perfect matching) or can be bound to another cysteine or free (unknown bonding state, yielding a non-perfect matching). The original set of constraints in [15] only focuses on complete structures (non extensible set or bases, in our terminology). It also has exponential size but the matching structure of the problem in that case allows the authors to derive a certificate formulation that reduces it to polynomial size. The MBG problem is not a matching problem but has the structure of a matroid and our formulation allows us to control the number of effectively enforced constraints by taking advantage of a greedy algorithm.

The idea of an online learning procedure that receives examples generated by an algorithm which constructs the output structure was inspired from the *Learning as Search Optimization* (LaSO) approach [4]. LaSO aims to solve a much broader class of structured output problems where good output structures can be generated by AI-style search algorithms such as beam search or A*. The generation of a fresh set of siblings in LaSO when the search is stuck with a frontier of wrong candidates (essentially a backtrack) is costly compared to our greedy selection procedure and (at least in principle) unnecessary when working on matroids.

Another general way to deal with the exponential growth of the search space is to introduce a generative model so that $\arg\max_y F(x, y)$ can be computed efficiently, e.g. by developing an appropriate dynamic programming algorithm. Stochastic grammars and related conditional models have been extensively used for this purpose [2]. These approaches work well if the generative model matches or approximates well the domain at hand. Unfortunately, as discussed in Section 2, the specific application problem we study in this paper cannot be even modeled by a context-free grammar. While we do not claim that it is impossible to devise a suitable generative model for this task (and indeed this is an interesting direction of research), we can argue that handling context-sensitiveness is hard. It is of course possible to approximate context sensitive dependencies using a simplified model. Indeed, an alternative view of the MBG problem is supervised sequence labeling, where the output string consists of symbols in $\mathcal{I}$. A (higher-order) hidden Markov model or chain-structured conditional random field could be used as the underlying generative model for structured output learning.

Unfortunately, these approaches are unlikely to be very accurate since models that are structured as linear chains of dependencies cannot easily capture long-ranged interactions such as those occurring in the example. In our preliminary experiments, $SVM_{HMM}$ [16] systematically assigned all bonded residues to the same ion, thus never correctly predicted the geometry except in trivial cases.

## 7 Conclusions

We have reported about the first successful solution to the challenging problem of predicting protein metal binding geometry from sequence alone. The result fills-in an important gap in structural and functional bioinformatics. Learning with structured outputs is a fairly difficult task and in spite of the fact that several methodologies have been proposed, no single general approach can effectively solve every possible application problem. The solution proposed in this paper draws on several previous ideas and specifically leverages the existence of a matroid for the metal binding problem. Other problems that formally exhibit a greedy structure might benefit of similar solutions.

### Acknowledgments

We thank Thomas Gärtner for very fruitful discussions.

## References

[1] M. Babor, S. Gerzon, B. Raveh, V. Sobolev, and M. Edelman. Prediction of transition metal-binding sites from apo protein structures. *Proteins*, 70(1):208–217, 2008.

[2] G. Bakir, T. Hofmann, B. Schölkopf, A. Smola, B. Taskar, and S. Vishwanathan, editors. *Predicting Structured Data*. The MIT Press, 2007.

[3] A. Bordes, S. Ertekin, J. Weston, and L. Bottou. Fast kernel classifiers with online and active learning. *Journal of Machine Learning Research*, 6:1579–1619, 2005.

[4] H. Daume III and D. Marcu. Learning as search optimization: Approximate large margin methods for structured prediction. In *Proc. of the 22nd Int. Conf. on Machine Learning (ICML'05)*, 2005.

[5] J. C. Ebert and R. B. Altman. Robust recognition of zinc binding sites in proteins. *Protein Sci*, 17(1):54–65, 2008.

[6] F. Ferrè and P. Clote. DiANNA 1.1: an extension of the DiANNA web server for ternary cysteine classification. *Nucleic Acids Res*, 34:W182–W185, 2006.

[7] P. Helman, B. M. E. Moret, and H. D. Shapiro. An exact characterization of greedy structures. *SIAM J. Disc. Math.*, 6(2):274–283, 1993.

[8] N. Hulo, A. Bairoch, V. Bulliard, L. Cerutti, B. A. Cuche, E. de Castro, C. Lachaize, P. S. Langendijk-Genevaux, and C. J. A. Sigrist. The 20 years of prosite. *Nucleic Acids Res*, 36:D245–9, 2008.

[9] E. L. Lawler. *Combinatorial Optimization: Networks and Matroids*. Holt, Rinehart and Winston, 1976.

[10] A. Messerschmidt, R. Huber, K. Wieghardt, and T. Poulos, editors. *Handbook of Metalloproteins*. John Wiley & Sons, 2004.

[11] A. Passerini, C. Andreini, S. Menchetti, A. Rosato, and P. Frasconi. Predicting zinc binding at the proteome level. *BMC Bioinformatics*, 8:39, 2007.

[12] A. Passerini, M. Punta, A. Ceroni, B. Rost, and P. Frasconi. Identifying cysteines and histidines in transition-metal-binding sites using support vector machines and neural networks. *Proteins*, 65(2):305–316, 2006.

[13] W. Shi, C. Zhan, A. Ignatov, B. A. Manjasetty, N. Marinkovic, M. Sullivan, R. Huang, and M. R. Chance. Metalloproteomics: high-throughput structural and functional annotation of proteins in structural genomics. *Structure*, 13(10):1473–1486, 2005.

[14] N. Shu, T. Zhou, and S. Hovmoller. Prediction of zinc-binding sites in proteins from sequence. *Bioinformatics*, 24(6):775–782, 2008.

[15] B. Taskar, V. Chatalbashev, D. Koller, and C. Guestrin. Learning structured prediction models: a large margin approach. *Proc. of the 22nd Int. Conf. on Machine Learning (ICML'05)*, pages 896–903, 2005.

[16] I. Tsochantaridis, T. Joachims, T. Hofmann, and Y. Altun. Large Margin Methods for Structured and Interdependent Output Variables. *The Journal of Machine Learning Research*, 6:1453–1484, 2005.

[17] J. Weston, O. Chapelle, A. Elisseeff, B. Scholkopf, and V. Vapnik. Kernel dependency estimation. *Advances in Neural Information Processing Systems*, 15:873–880, 2003.
